# Image Reconstruction by Linear Programming

**Koji Tsuda**[*†] **and Gunnar Rätsch**[*‡]
[*]Max Planck Institute for Biological Cybernetics
Spemannstr. 38, 72076 Tübingen, Germany
[†]AIST CBRC, 2-43 Aomi, Koto-ku, Tokyo, 135-0064, Japan
[‡]Fraunhofer FIRST, Kekuléstr. 7, 12489 Berlin, Germany
{koji.tsuda,gunnar.raetsch}@tuebingen.mpg.de

## Abstract

A common way of image denoising is to project a noisy image to the subspace of admissible images made for instance by PCA. However, a major drawback of this method is that all pixels are updated by the projection, even when only a few pixels are corrupted by noise or occlusion. We propose a new method to identify the noisy pixels by $\ell_1$-norm penalization and update the identified pixels only. The identification and updating of noisy pixels are formulated as *one* linear program which can be solved efficiently. Especially, one can apply the $\nu$-trick to directly specify the fraction of pixels to be reconstructed. Moreover, we extend the linear program to be able to exploit prior knowledge that occlusions often appear in contiguous blocks (e.g. sunglasses on faces). The basic idea is to penalize boundary points and interior points of the occluded area differently. We are able to show the $\nu$-property also for this extended LP leading a method which is easy to use. Experimental results impressively demonstrate the power of our approach.

## 1 Introduction

Image denoising is an important subfield of computer vision, which has extensively been studied (e.g. [2, 6, 1, 9]). The aim of image denoising is to restore the image corrupted by noise as close as possible to the original one. When one does not have any prior knowledge about the distribution of images, the image is often denoised by simple smoothing (e.g. [2, 1]). When one has a set of template images, it is preferable to project the noisy image to the linear manifold made by PCA, which is schematically illustrated in Fig. 1 (left). One can also construct a nonlinear manifold, for instance by kernel PCA, requiring additional computational costs [6]. The projection amounts to finding the closest point in the manifold according to some distance. Instead of using the standard Euclidean distance (i.e. the least squares projection), one can adopt a robust loss such as Huber's loss as the distance, which often gives a better result (robust projection [9]). However, a major drawback of these projection approaches is that all pixels are updated by the projection. However, typically only a few pixels are corrupted by noise, thus non-noise pixels should best be left untouched.

This paper proposes a new denoising approach by linear programming, where the $\ell_1$-norm regularizer is adopted for automatic identification of noisy pixels – only these are updated. The identification and updating of noisy pixels are neatly formulated as one linear program. The theoretical advantages of linear programming lie in duality and optimality conditions. By considering both primal and dual problems at the same time, one can construct effective and highly principled optimizers such as interior point methods. Also, the optimality con-

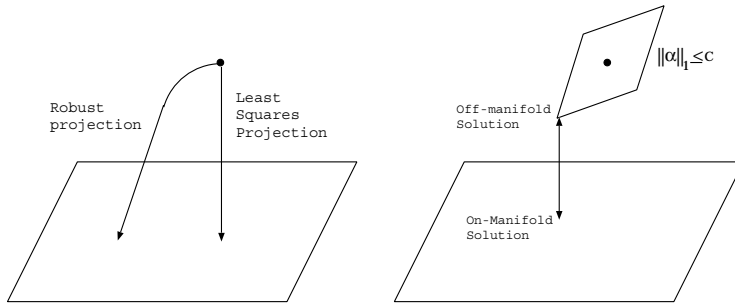

**Figure 1:** Difference between projection methods (left) and our LP method (right).

ditions enables us to predict important properties of the optimal solution before we actually solve it. In particular, we can explicitly specify the fraction of noisy pixels by means of the $\nu$-trick originally developed for SVMs [8] which was later applied to Boosting [7].

In some cases the noisy pixels are not scattered over the image ("impulse noise"), but form a considerably large connected region ("block noise"), e.g. face images occluded by sunglasses. By using the prior knowledge that the noisy pixels form blocks, we should be able to improve the denoising performance. Several ad-hoc methods have been proposed so far (e.g. [9]), but we obviously need a more systematic way. We will show that a very simple modification of the linear program has the effect that we can control how block-shape like the identified and reconstructed region is. In the experimental section we will show impressive results on face images from the MPI face data base corrupted by impulse and block noises.

## 2  Image Denoising by Linear Programming

Let $\{\boldsymbol{t}_j\}_{j=1}^J$ be the set of vectors in $\Re^N$, which have been derived for instance by principal component analysis. The linear manifold of admissible images is described as

$$\mathcal{T} = \left\{ \boldsymbol{t} \mid \boldsymbol{t} = \sum\nolimits_{j=1}^J \beta_j \boldsymbol{t}_j, \beta_j \in \Re \right\}$$

Now we would like to denoise a noisy image $\boldsymbol{x} \in \Re^N$. Let us describe the denoised image as $\bar{\boldsymbol{x}}$. In order that the denoised image $\bar{\boldsymbol{x}}$ is similar to admissible images, $\bar{\boldsymbol{x}}$ should be close to the manifold:

$$\min_{\boldsymbol{\beta}} d_1\left(\bar{\boldsymbol{x}}, \sum\nolimits_{j=1}^J \beta_j \boldsymbol{t}_j\right) \leq \epsilon_1, \tag{1}$$

where $d_1$ is a distance between two images. Also, we have to constrain $\boldsymbol{x}$ to be close to $\bar{\boldsymbol{x}}$, otherwise the denoised image becomes completely independent from the original image:

$$d_2(\bar{\boldsymbol{x}}, \boldsymbol{x}) \leq \epsilon_2, \tag{2}$$

where $d_2$ is another distance. A number of denoising methods can be produced by choosing different distances and changing how to minimize the two competing objectives (1) and (2). In projection methods, $\epsilon_1$ is simply set to zero and $\epsilon_2$ is minimized with $d_2$ being set to the Euclidean distance or a robust loss.

**A Linear Programming Formulation**  Our wish is that most pixels of $\boldsymbol{x}$ stay unchanged in $\bar{\boldsymbol{x}}$, in other words, the difference vector $\boldsymbol{\alpha} = \bar{\boldsymbol{x}} - \boldsymbol{x}$ should be *sparse*. For this purpose, $d_2$ is chosen as the $\ell_1$-norm, as it is well known that the $\ell_1$-norm constraints produce sparse solutions (e.g. [7]). Also for $d_1$, the $\ell_\infty$-norm is especially interesting as it leads to linear programming. We design the optimization problem as follows:

$$\min_{\boldsymbol{\alpha},\boldsymbol{\beta}} \quad \left\| \boldsymbol{x} + \boldsymbol{\alpha} - \sum\nolimits_{j=1}^J \beta_j \boldsymbol{t}_j \right\|_\infty \tag{3}$$

$$\|\boldsymbol{\alpha}\|_1 \leq C, \tag{4}$$

where $\|x\|_\infty = \max_i |x_i|$, $\|\alpha\|_1 = \sum_{i=1}^N |\alpha_i|$ and $C$ is a constant to determine the sparseness, i.e. the solution $\alpha$ tends to become more sparse as $C$ decreases. Geometrically, this optimization problem is explained as Fig. 1 (right). The constraint (4) keeps $\bar{x}$ within the $\ell_1$-sphere centered on $x$. The optimization finds a point in the sphere, which is closest to the linear manifold. As a side effect, we have another solution $\sum_j \beta_j t_j$ on the manifold. We call the former the "off-manifold solution" and the latter "on-manifold solution". Here, we are mainly concerned with the off-manifold solution, because of the sparsity.

Let us actually formulate (3) as a linear programming problem. It is equivalent to

$$\min_{\alpha,\beta,\epsilon} \quad \frac{1}{N}\sum_{n=1}^N |\alpha_n| + \nu\epsilon \tag{5}$$

$$\left| x_n + \alpha_n - \sum_{j=1}^J \beta_j t_{jn} \right| \leq \epsilon, \quad n = 1,\dots,N,$$

where $\nu$ is a regularization parameter. Still this problem is not linear programming because of $|\alpha_n|$ in the objective function. Next let us restate $\alpha$ as follows:

$$\alpha = \alpha^+ - \alpha^-, \quad \alpha_n^+, \alpha_n^- \geq 0, \quad n = 1,\dots,N.$$

Then (5) is rewritten as the following linear programming problem:

$$\min_{\alpha^\pm,\beta,\epsilon} \quad \frac{1}{N}\sum_{n=1}^N (\alpha_n^+ + \alpha_n^-) + \nu\epsilon \tag{6}$$

$$\alpha_n^+, \alpha_n^- \geq 0, \quad \left| x_n + \alpha_n^+ - \alpha_n^- - \sum_{j=1}^J \beta_j t_{jn} \right| \leq \epsilon, \quad n = 1,\dots,N. \tag{7}$$

Here we used the well known fact that either $\alpha_n^+$ or $\alpha_n^-$ is zero at the optimum.

**The $\nu$-Trick**  In the above optimization problem, the regularization constant $\nu$ should be determined to control the fraction of updated pixels. Interestingly, $\nu$ has an intuitive meaning as follows: Let $N_p$ denote the number of nonzero elements in $\alpha$. Furthermore let $N_c$ be the number of "crucial pixels" which are not updated, but the corresponding constraint constraints (7) are met as equalities. If one of these pixels is modified, then it will likely lead to a different solution, while changing any of the other $N - N_p - N_c$ pixels locally does not change the optimal solution.

**Proposition 1.** *Suppose the optimal $\epsilon$ is greater than $0$. Then the number of nonzero elements $N_p$ in the optimal $\alpha$ is*

1. *upper bounded by $\nu N$, i.e. $N_p \leq \nu N$ and*

2. *lower bounded by $\nu N - N_c$, i.e. $N_p \geq \nu N - N_c$.*

The proof is a special case of the one of Proposition 2 and is omitted. The slack in the bound only comes from $N_c$. In practice we usually observed small values of $N_c$. We suspect that its value is related to $J$ – the number of basis vectors.

In terms of images, one can bound the anticipated fraction of noise pixels by $\nu$. In contrast, the constant $C$ in (4) specifies the sum of noise magnitudes, which is in practice rather difficult to figure out.

## 3   Dealing with Block Noises

**Preliminaries**  When noises are clustered as blocks, this prior knowledge is considered to lead to an increased denoising performance. So far we could only control the number of modified pixels which corresponds to the area of reconstruction. In this section we also consider the length of the boundary of the identified pixels. For instance, consider the three occlusion patterns in Figure 2. The pixel is white, when it is identified as noisy/occluded and black otherwise. In the first case (left) the occlusion forms a block, in the second case the letters "lp" and in the third case the pixels are randomly distributed. The covered area is the same for all three cases.

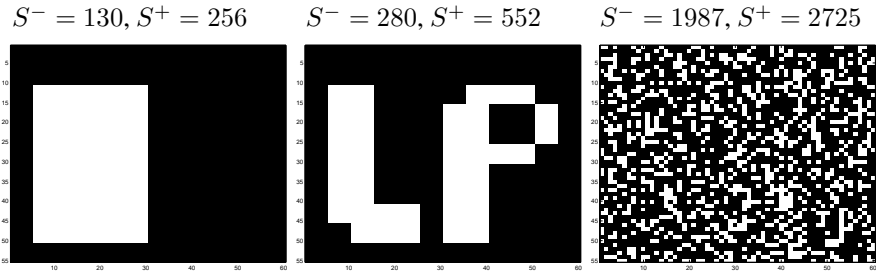

$S^- = 130, S^+ = 256$  $S^- = 280, S^+ = 552$  $S^- = 1987, S^+ = 2725$

**Figure 2:** Three occlusion patterns with different degrees of having a block shape.

We will now define two measures of how much an occlusion pattern *mismatches* the block shape. It is related to the length of the boundary. Note that optimal "block" shapes have shortest boundaries. (It depends on the metric what will be optimal.)

We distinguish between two types of penalties: first, the ones which occur when a reconstructed pixel is a neighbor of an untouched pixel ("boundary point") and second, if a reconstructed pixel is neighbor of another such pixel, but the corrections are in different directions ("inversion point"). We have two definitions for our scores, which we will later relate to the solution of our extended linear program. The differences between the two scores $S^-$ and $S^+$ are only in subtle details in how to count boundary points and inversion points:

- Let $N_b^-$ be the number of pixels $n$ which satisfy: (a) $\alpha_n = 0$ and there exists $m \in G(n)$ such that $\alpha_m \neq 0$ (*outer boundary point*) or (b) $\alpha_n \neq 0$ and for *all* $m \in G(n)$ holds $\alpha_m = 0$ (*single pixel change*). Let $N_i^-$ be the number of pixels $n$ with $\alpha_n \alpha_m < 0$ for at least one $m \in G(n)$ and $\alpha_n \alpha_m \leq 0$ for all $m \in G(n)$ (*single inversion point*). The first score is computed as $S^- := N_b^- + 2N_i^-$.

- Let $N_b^+$ be the number of pixels $n$ which satisfy: (a) $\alpha_n = 0$ and there exists $m \in G(n)$ such that $\alpha_m \neq 0$ (*outer boundary point*) or (b) $\alpha_n \neq 0$ and there exists $m \in G(n)$ with $\alpha_m = 0$ (*inner boundary point*). Let $N_i^+$ be the number of pixels $n$ with $\alpha_n \alpha_m < 0$ for at least one $m \in G(n)$ (*inversion point*). Then the second score is computed as $S^+ := N_b^+ + 2N_i^+$.

The main difference between the two scores is that $S^+$ counts the length of the inner *and* outer boundary, while $S^-$ only counts the outer boundary.

**The Extended LP**  The question is how we can introduce these definitions into a linear program, which somehow penalizes these scores. As we will show in the following proposition, it turns out that it is enough to penalize the differences between neighboring $\alpha$'s. We introduce a new set of variables (the $\gamma$'s) which account for these differences and which are linearly penalized. We control the contribution of the $\gamma$'s with the one of the $\alpha$'s by introducing a new parameter $\lambda \in (0, 1)$ – if $\lambda = 0$, then the original LP is recovered:

$$\min_{\gamma \geq 0, \alpha, \epsilon \geq 0, \beta} \quad \frac{\lambda}{N} \sum_{n=1}^{N} \gamma_n + \frac{1-\lambda}{N} \sum_{n=1}^{N} |\alpha_n| + \nu \epsilon \tag{8}$$

$$\left| x_n + \alpha_n - \sum_{j=1}^{J} \beta_j t_{j,n} \right| \leq \epsilon \quad \text{for all } n = 1, \ldots, N$$

$$|\alpha_n - \alpha_m| \leq \gamma_n \quad \text{for all } m \in G(n)$$

We will show in the experimental part that these novel constraints lead to substantial improvements for block noises. The analysis of this linear program is considerably more difficult than of the previous one. However, we will show that the $\nu$-trick still works in a

generalized manner with some subtleties. We will show in the following Proposition that LP (8) trades-off the area $N_p$ with the penalty scores $S^-$ and $S^+$:

**Proposition 2.** *Let $N_c$ the number of crucial pixels and $N_p$ the number of updated pixels (as before). Assume the optimal $\epsilon$ is greater $0$. Then holds:*

1. *The $\lambda$-weighted average between area of the occlusion and score $S^-$ is not greater than $\nu N$, i.e.*
$$(1 - \lambda)N_p + \lambda S^- \leq \nu N \qquad (9)$$

2. *If $\lambda < \frac{1}{2+|G|}$, then the $\lambda$-weighted average between area of the occlusion and score $S^+$ is not smaller than $\nu N$ minus $2N_c$, i.e.*
$$(1 - \lambda)N_p + \lambda S^+ > \nu N - 2N_c, \qquad (10)$$
*where $|G| := \max_n |G(n)|$*

Note that the slackness in (10) again only comes from the number of crucial points $N_c$. The restriction $\lambda < \frac{1}{2+|G|}$ only concerns the second part and and not the functioning of the LP in practice. It can be made less restrictive, but this goes beyond the scope of this paper. Due to space limitations we have to omit the proof. It is found in a technical report, which can be downloaded from `http://www.kyb.tuebingen.mpg.de/publications/pdfs/pdf2420.pdf`.

## 4 Denoising by QP and Robust Statistics

A characteristic of the LP method is that the $\ell_\infty$-norm is used as $d_1$. But other choices are of course possible. For example, when the squared loss is adopted as $d_1$, the optimization problem (3) is rewritten as

$$\min_{\boldsymbol{\alpha},\boldsymbol{\beta}} \frac{1}{N} \sum_{n=1}^{N} \left( x_n + \alpha_n - \sum_{j=1}^{J} \beta_j \boldsymbol{t}_{jn} \right)^2 + \nu |\alpha_n|. \qquad (11)$$

This is a quadratic program (QP), which can also be solved by standard algorithms. In our experience, QP takes longer time to solve than LP and the denoising performance is more or less the same. Furthermore the $\nu$-trick does not hold for QP. Nevertheless, it is interesting to take a close look at the QP method as it is more related to existing robust statistical approaches [2, 9]. The QP can partially be solved analytically with respect to $\boldsymbol{\alpha}$:

$$\min_{\boldsymbol{\beta}} \sum_{n=1}^{N} \rho\left( x_n - \sum_{j=1}^{J} \beta_j \boldsymbol{t}_{jn} \right), \qquad (12)$$

where $\rho$ is the Huber's loss

$$\rho(t) = \begin{cases} \frac{t^2}{N} & -\frac{N\nu}{2} \leq t \leq \frac{N\nu}{2} \\ |t| - \frac{N\nu^2}{4} & \text{otherwise.} \end{cases}$$

Thus, the on-manifold solution of (11) corresponds to the robust projection by the Huber's loss. In other words, $\boldsymbol{\alpha}$ is considered as a set of *slack variables* in the robust projection. It is worthwhile to notice another choice of slack variables proposed in [2]:

$$\min_{\boldsymbol{z},\boldsymbol{\beta}} \frac{1}{2\gamma} \sum_{n=1}^{N} z_n \left( x_n - \sum_{j=1}^{J} \beta_j \boldsymbol{t}_{jn} \right)^2 + \gamma \frac{1}{2z_n}. \qquad (13)$$
$$0 \leq z_n \leq 1, \ n = 1, \dots, N.$$

Here the slack variables are denoted as $\boldsymbol{z}$, which is called the *outlier process* [2]. Notice $\gamma$ is a regularization constant. Let us define $g_n = x_n - \sum_{j=1}^{J} \beta_j \boldsymbol{t}_{jn}$. Then the inside problem with respect to $z_n$ can be analytically solved, and we have the reduced problem as

$$\min_{\boldsymbol{\beta}} \sum_{n=1}^{N} h_\gamma \left( x_n - \sum_{j=1}^{J} \beta_j \boldsymbol{t}_{jn} \right) \qquad (14)$$

where $h_\gamma(t)$ is again the Huber's loss function: $h_\gamma(t) = \frac{t^2}{2\gamma} + \frac{\gamma}{2}$ if $|t| < \gamma$ and $|t|$ if $|t| \geq \gamma$. The outlier process tells one which pixels are ignored, but it does not directly represent the denoised image. From the viewpoint of denoising, our slack variables $\boldsymbol{\alpha}$ seem to make more sense.

# 5 Experiments

We applied our new methods and the standard methods to the MPI face database [3, 4]. This dataset has 200 face images (100 males and 100 females) and each image is rescaled to 44×64. The images are artificially corrupted by impulse and block noises. As impulse noises, 20% of the pixels are chosen randomly and set to 0. For block noises, a rectangular region (10% of the pixels) is set to zero to hide the eyes. We hide the same position for all images, but the position of the rectangle is *not* known to our algorithm. The task is to recover the original image based on the remaining 199 images (i.e. l.o.o. cross validation).

Our linear program is compared against the least squares projection and the robust projection using Huber's loss (i.e. the on-manifold solution of QP). One could also apply the non-convex robust losses for better robustness, e.g. Tukey's biweight, Hampel, Geman-McClure, etc [2]. On the other hand, we could also use the non-convex regularizers which are "steeper" than the $\ell_1$-norm for greater sparsity [5]. However, we will not trade convexity with denoising performance here, because local minima often put practitioners into trouble. As a reference, we also consider an *idealistic* denoising method, to which we give the true position of noises. Here, the pixel values of noisy positions are estimated by the least squares projection only with respect to the non-noise pixels. Then, the estimated pixel values are plugged back into the original image. The linear manifold is made by PCA from the remaining 199 images. The number of principal components is determined such that the idealistic method performs the best. For impulse and block noise images, it turned out to be 110 and 30, respectively.

The reconstruction errors of LP and QP for impulse noises are shown in Fig. 4. Here, the reconstruction error is measured by the $\ell_2$-norm between the images. Also an example of denoising is shown in Fig. 3. Both in LP and QP, the off-manifold solution outperforms

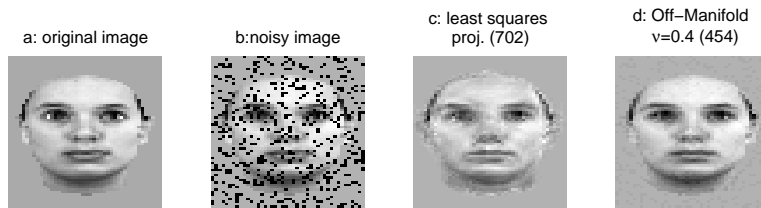

a: original image    b:noisy image    c: least squares proj. (702)    d: Off–Manifold v=0.4 (454)

**Figure 3:** A typical result of denoising impulse noise. (a) An original face image. (b) The image corrupted by impulse noise. (c) Reconstruction by the least squares projection to the PCA basis. The number in (·) shows the reconstruction error. (d) Reconstruction by the LP (off-m.) when $\nu = 0.4$.

the on-manifold one, which confirms our intuition that it is effective to keep most pixels unchanged. Compared with the least squares projection, the difference is so large that one can easily see it in the reconstructed images (Fig. 3). Notably, the off-manifold solutions of LP and QP (cf. the solid curves in Fig. 4, left and right) performed significantly better than the on-manifold solution of QP, which corresponds to the robust projection using Huber's loss (cf. the dashed curve in Fig. 4 right).

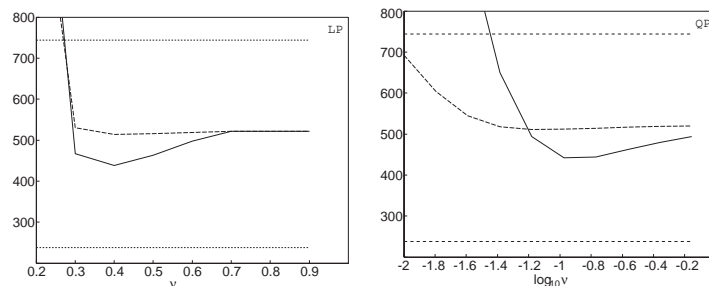

**Figure 4:** Reconstruction errors of LP and QP methods for impulse noise. The solid and dashed lines corresponds to the off-manifold and on-manifold solutions. The flat lines correspond to the least squares projection and the unrealistic setting where the correct positions of noises are given.

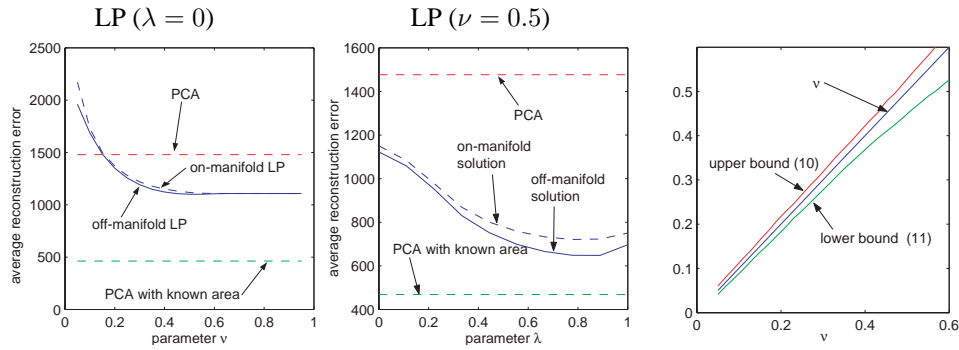

**Figure 5:** Reconstruction errors of the LP method for block noises. (Left) the reconstruction error of the "plain" LP, where the block constraints are not taken into account ($\lambda = 0$). The right plot shows the improvement for increased $\lambda$ and fixed $\nu = 1/2$.

**Figure 6:** Illustration of Prop. 2: For $\lambda = 0.15$ we compute the lower and upper bound of $\nu N$ for different $\nu$'s.

The results for block noises are shown in Fig. 5, where we again averaged over the 200 faces (using l.o.o. cross validation for the construction of the PCA basis). In the left figure, we measure the reconstruction error for various $\nu$'s with fixed $\lambda = 0$, i.e. the block constraints are not taken into account. As in the case with impulse noise, the error is smaller than that of the least squares regression (PCA projection), and the minimum is attained around $\nu = 1/2$. Moreover, we investigated how the error is further reduced by increasing $\lambda$ from 0. As shown in the right figure, we obtain a significant improvement. Actually, there is not much room for improvements, since even the idealistic case where the position of the occlusion is know is not much better.

An example of reconstructed images are shown in Fig. 7. Here we have shown variables $\alpha$ and $\gamma$ as well. When $\lambda = 0$, nonzero $\alpha$'s appear not only in occluded part but also for instance along the face edge (Fig. 7:e). When $\lambda = 1/2$, nonzero $\alpha$'s are more concentrated in the occluded part, because the block constraints suppress a isolated nonzero values (Fig. 7:h). In Fig. 7:i, one can see high $\gamma$'s in the edge pixels of occluded region, which indicates that the block constraints are active for those pixels.

Finally we empirically verify Proposition 2. In Fig. 6 we plot the lower and upper bound of $\nu$ as given in Proposition 2 for different values of $\nu$. Observe that the difference between lower and upper bound is quite small.

## 6   Concluding Remarks

In summary, we have presented a new image denoising method based on linear programming. Our main idea is to introduce sparsity by detaching the solution slightly from the manifold. The on-manifold solution of our method is related to existing robust statistical approaches. Remarkably, our method can deal with block noises while retaining the convexity of the optimization problem (every linear program is convex). Existing approaches (e.g. [9]) tend to rely on non-convex optimization to include the prior knowledge that the noises form blocks. Perhaps surprisingly, our convex approach can solve this problem to a great extent. We are looking forward to apply the linear programming to other computer vision problems which involve combinatorial optimization, e.g. image segmentation. Also, it is interesting to explore the limitations of convex optimization, since – naturally – convex optimization cannot solve every problem. Nevertheless, according to our experience in this work, we feel that the power of convex optimization is not fully exploited.

**Acknowledgment**   The authors gratefully acknowledge A. Graf for preparing the face image dataset. We would like to thank B. Schölkopf, J. Weston, T. Takahashi, T. Kurita, S. Akaho and Chan-Kyoo Park for fruitful discussions.

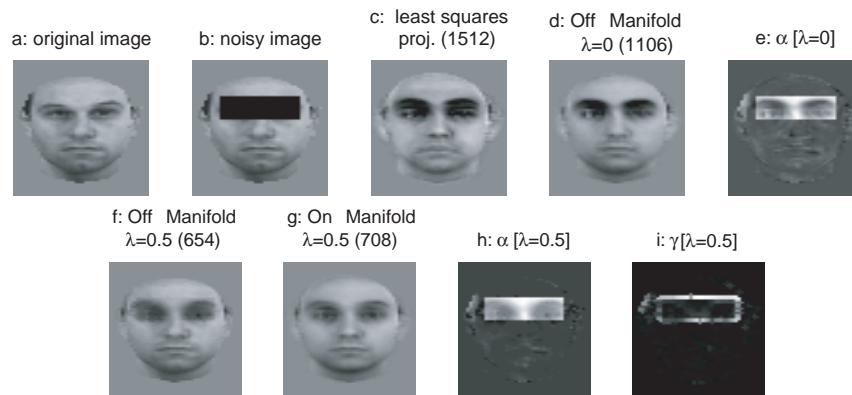

**Figure 7:** A typical result of denoising block noises ($\nu = 0.5$). The numbers in ($\cdot$) in (c),(d),(f),(g) show the reconstruction errors. The image (d) shows the denoising result when the block constraints are not taken into account ($\lambda = 0$, $\nu = 1/2$). This result improves by imposing the block constraints ($\lambda = 1/2$, $\nu = 1/4$) as shown in (f) and (g), which are the off and on-manifold solutions, respectively. The images (e),(h) and (i) show the parameter values obtained as the result of linear programming (see the text for details).

## References

[1] A. Ben Hamza and H. Krim. Image denoising: A nonlinear robust statistical approach. *IEEE Trans. Signal Processing*, 49(12):3045–3054, 2001.

[2] M.J. Black and A. Rangarajan. On the unification of line processes, outlier rejection, and robust statistics with applications in early vision. *International Journal of Computer Vision*, 25(19):57–92, 1996.

[3] V. Blanz and T. Vetter. A morphable model for the synthesis of 3D faces. In *SIGGRAPH'99 Conference Proceedings*, pages 187–194, 1999.

[4] A.B.A. Graf and F.A. Wichmann. Gender classification of human faces. In H.H. Bülthoff, S.-W. Lee, T.A. Poggio, and C. Wallraven, editors, *Biologically Motivated Computer Vision 2002*, LNCS 2525, pages 491–501, 2002.

[5] O.L. Mangasarian. Machine learning via polyhedral concave minimization. Technical Report 95-20, Computer Sciences Department, University of Wisconsin, 1995.

[6] S. Mika, B. Schölkopf, A.J. Smola, K.-R. Müller, M. Scholz, and G. Rätsch. Kernel PCA and de–noising in feature spaces. In M.S. Kearns, S.A. Solla, and D.A. Cohn, editors, *Advances in Neural Information Processing Systems*, volume 11, pages 536–542. MIT Press, 1999.

[7] G. Rätsch, B. Schölkopf, A.J. Smola, S. Mika, T. Onoda, and K.-R. Müller. Robust ensemble learning. In A.J. Smola, P.L. Bartlett, B. Schölkopf, and D. Schuurmans, editors, *Advances in Large Margin Classifiers*, pages 207–219. MIT Press, Cambridge, MA, 2000.

[8] B. Schölkopf, A. Smola, R.C. Williamson, and P.L. Bartlett. New support vector algorithms. *Neural Computation*, 12:1207 – 1245, 2000. also NeuroCOLT Technical Report NC-TR-1998-031.

[9] T. Takahashi and T. Kurita. Robust de-noising by kernel PCA. In J.R. Dorronsoro, editor, *Artificial Neural Networks – ICANN 2002*, LNCS 2415, pages 727–732. Springer Verlag, 2002.
